# Dirichlet-Bernoulli Alignment: A Generative Model for Multi-Class Multi-Label Multi-Instance Corpora

**Shuang-Hong Yang**
College of Computing
Georgia Tech
shy@gatech.edu

**Hongyuan Zha**
College of Computing
Georgia Tech
zha@cc.gatech.edu

**Bao-Gang Hu**
NLPR & LIAMA
Chinese Academy of Sciences
hubg@nlpr.ia.ac.cn

## Abstract

We propose *Dirichlet-Bernoulli Alignment* (DBA), a generative model for corpora in which each pattern (e.g., a document) contains a set of instances (e.g., paragraphs in the document) and belongs to multiple classes. By casting predefined classes as latent Dirichlet variables (i.e., instance level labels), and modeling the multi-label of each pattern as Bernoulli variables conditioned on the weighted empirical average of topic assignments, DBA automatically aligns the latent topics discovered from data to human-defined classes. DBA is useful for both pattern classification and instance disambiguation, which are tested on text classification and named entity disambiguation in web search queries respectively.

## 1 Introduction

We consider multi-class, multi-label and multi-instance classification ($M^3C$), a task of learning decision rules from corpora in which each pattern consists of multiple instances[1] and is associated with multiple classes. $M^3C$ finds its application in many fields: For example, in web page classification, a web page (*pattern*) typically comprises of different entities (*instances*) (e.g., texts, pictures and videos) and is usually associated with several different topics (e.g., finance, sports and politics). In such tasks, a pattern usually consists of a set of instances, and the possible instances may be too diverse in nature (e.g., of different structures or types, described by different features) to be represented in a universal space. What makes the problem more complicated and challenging is that the pattern is usually ambiguous, i.e., it can belong to several different classes simultaneously. Traditional classification algorithms are typically incapable of handling such complications.

Even for corpora consisting of relatively homogenous data, treating the tasks as $M^3C$ might still be advantageous since it enables us to explore the inner structures and the ambiguity of the data simultaneously. For example, in text classification, a document usually comprises several separate semantic parts (e.g., paragraphs), and several different topics are evolving along these parts. Since the class-labels are often only locally tied to the document (e.g., paragraphs are often far more topic-focused than the whole document), base the classification on the whole document would incur too much noise and in turn harm the performance. In addition, treating the task as $M^3C$ also offers a natural way to track the topic evolution along paragraphs, a task that is otherwise difficult to handle.

$M^3C$ also arises naturally when the acquisition of labeled data is expensive. For example, in scene classification, a picture usually contains several objects (e.g., cat, desk, man) belonging to several different classes (e.g., animal, furniture, human). Ideal annotation requires a skilled expert to specify both the exact location and class label of each object in the image, which, though not completely impossible, involves too much human efforts especially for large image repositories. The annotation burden would be greatly relieved if each image is labeled as a whole (e.g., a caption indicating what is in the image), which, however, requires the learning system to be capable to handle $M^3C$ tasks.

Recently, the Latent Dirichlet Allocation (LDA, [4]) model has been established for automatic extraction of topical structures from large repository of documents. LDA is a highly-modularized probabilistic model with various variations and extensions (e.g., [2, 3]). By modeling a document as a mixture over topics, LDA allows each document to be associated with multiple topics with different proportions, and thus provides a promising way to capture the heterogeneity/ambiguity in the data. However, the topics discovered by LDA are implicit (i.e., each topic is expressed as a distribution over words, comprehensible interpretation of which requires human expertise), and cannot be easily aligned to the topics of human interests. In addition, the standard LDA does not model the multi-instance structure of a pattern. Hence, LDA and its like cannot be directly applied to $M^3C$.

In this paper, by taking advantage of the LDA building blocks, we present a new probabilistic generative model for multi-class, multi-label and multi-instance corpora, referred to as *Dirichlet-Bernoulli Alignment* (DBA). DBA assumes a tree-structure about the data, i.e., each multi-labeled pattern is a bag of single-labeled instances. In DBA, each pattern is modeled as a mixture over the set of predefined classes, an instance is then generated independently conditioned on a sampled class-label, and the label of a pattern is generated from a Bernoulli distribution conditioned on all the sampled labels used for generating its instances. DBA is essentially a topic model similar to LDA except that (1) an instance rather than a single feature is generated conditioned on each sampled topic; and (2) instead of using implicit topics for dimensionality reduction as in LDA, DBA casts each class as an explicit topic to gain discriminative power from the data. Through likelihood maximization, DBA automatically aligns the topics discovered from the data to the predefined classes of our interests. DBA can be naturally tailored to $M^3C$ tasks for both pattern classification and instance disambiguation. In this paper, we apply the DBA model to text classification tasks and an interesting real-world problem, i.e., named entity disambiguation for web search queries. The experiments confirm the usefulness of the proposed DBA model.

The rest parts of this paper is organized as follows. Section 2 briefly reviews some related topics and Section 3 presents the formal description of the corpora used in $M^3C$ and the basic assumptions of our model. Section 4 introduces the detailed DBA model. In Section 5, we establish algorithms for inference and parameter estimation for DBA. And in Section 6, we apply the DBA model to text classification and query disambiguation tasks. Finally, Section 7 presents concluding remarks.

## 2 Related Works

Traditional classification largely focuses on a single-label single-instance framework (i.e., $i.i.d$ patterns, associated with exclusive/disjoint classes). However, the real-world is more like a web of (sub-)patterns connected with a web of classes that they belong to. Clearly, $M^3C$ reflects more of the reality. Recently, two partial solutions, i.e., multi-instance classification (MIC) [7, 11, 1] and multi-label classification (MLC) [10, 8, 5] were investigated. MIC assumes that each pattern consists of multiple instances but belongs to a single class, whereas MLC studies single-instance pattern associated with multiple classes. Although both MLC and MIC have drawn increasing attentions in the literature, neither of them can handle the cases where multi-instance and multi-label are simultaneously present. Perhaps the first work investigating $M^3C$ is [13], in which the authors proposed an *indirect* solution, i.e., to convert an $M^3C$ task into several MIC or MLC sub-tasks each of which is then divided into single-label and single-instance classification problems and solved by discriminative algorithms such as AdaBoost or SVM. A practical challenge of this approach is its complexity, i.e, the number of sub-tasks can be huge, making the training data extremely sparse for each sub-classifier and the computation cost unacceptably high in both training and testing. Recently, Cour et al proposed a discriminative framework [6] based on convex surrogate loss minimization for classifying ambiguously labeled images; and Xu et al established a hybrid generative/discriminative approach (i.e., a heuristically regularized LDA classifier) [12] to mining named entity from web search click-through data. In this paper, we present a *generative* approach for $M^3C$.

Our proposed DBA model can be viewed as a supervised version of topic models. A widely used topic model for categorical data is the LDA model [4]. By modeling a pattern as a random mixture over latent topics and a topic as a Multinomial distribution over features in a dictionary, LDA is effective in discovering implicit topics from a corpus. The supervised LDA (sLDA) model [2], by linking the empirical topics to the label of each pattern, is able to learn classifiers using Generalized Linear Models. However, both LDA and sLDA are in essence dimensionality reduction techniques, and cannot be employed directly for the $M^3C$ tasks.

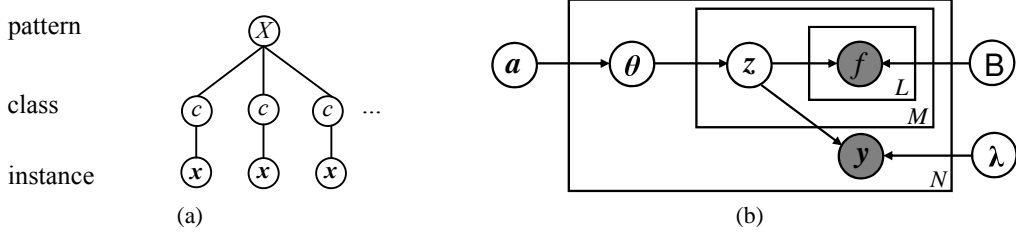

Figure 1: (a): Tree structure of a multi-class multi-label multi-instance corpus. (b):A graphic representation of the DBA model with multinomial bag-of-feature instance model.

## 3 Problem Formalization

Intuitively, we can think of a pattern as a document, an instance as a paragraph, and a feature as a word. In M³C, we are interested in inferring class labels for both the document and its paragraphs.

Formally, let $\mathcal{X} \subset \mathbb{R}^D$ denote the instance space (e.g., a vector space), $\mathcal{Y} = \{1, 2, \ldots, C\}$ $(C > 2)$ the set of class labels, and $\mathcal{F} = \{f_1, f_2, \ldots, f_D\}$ the dictionary of features. A multi-class, multi-label multi-instance corpus $\mathcal{D}$ consists of a set of input patterns $\{X_n\}_{n=1,2,\ldots,N}$ along with the corresponding labels $\{Y_n\}_{n=1,2,\ldots,N}$, where each pattern $X_n = \{\mathbf{x}_{mn}\}_{m=1,2,\ldots,M_n}$ contains a set of instances $\mathbf{x}_{mn} \in \mathcal{X}$, and $Y_n \subset \mathcal{Y}$ consists of a set of class labels. The goal of M³C is to find a decision rule $Y = \varphi(X) : 2^{\mathcal{X}} \to 2^{\mathcal{Y}}$, where $2^{\mathcal{A}}$ denotes the power set of a set $\mathcal{A}$. For simplicity, we make the following assumptions.

*Assumption 1* [Exchangeability]: A corpus is a bag of patterns, and each pattern is a bag of instances.

*Assumption 2* [Distinguishablity]: Each pattern can belong to several classes, but each instance belongs to a single class.

These assumptions are equivalent to assuming a tree structure for the corpus (Figure 1(a)).

## 4 Dirichlet-Bernoulli Alignment

In this section, we present Dirichlet-Bernoulli Alignment (DBA), a probabilistic generative model for the multi-class, multi-label and multi-instance corpus described in Section 3. In DBA, each pattern $X$ in a corpus $\mathcal{D}$ is assumed to be generated by the following process:

1. Sample $\theta \sim \text{Dir}(\mathbf{a})$.
2. For each of the $M$ instances in $X$:
   ▷ Choose a class $\mathbf{z} \sim \text{Mult}(\theta)$;
   ▷ Generate an instance $\mathbf{x} \sim p(\mathbf{x}|\mathbf{z}, B)$;
3. Generate the label $\mathbf{y} \sim p(\mathbf{y}|\mathbf{z}_{1:M}, \boldsymbol{\lambda})$.

We assume the total number of predefined classes, $C$, is known and fixed. In DBA, $\mathbf{a} = [a_1, \ldots, a_C]^\top$ with $a_c \geqslant 0, c = 1, \ldots, C$, is a $C$-vector prior parameter for a Dirichlet distribution $\text{Dir}(\mathbf{a})$, which is defined in the $(C\text{-}1)$-simplex: $\theta_c \geqslant 0, \sum_{c=1}^C \theta_c = 1$. $\mathbf{z}$ is a class indicator, i.e., a binary $C$-vector with the 1-of-$C$ code: $z_c = 1$ if the $c$-th class is chosen, and $\forall i \neq c, z_i = 0$. $\mathbf{y} = [y_1, \ldots, y_C]^\top$ is also a binary $C$-vector with $y_c = 1$ if the pattern $X$ belongs to the $c$-th class and $y_c = 0$ otherwise.

In this paper, we assume the label of a pattern is generated by a cost-sensitive voting process according to the labels of the instances in it, which is intuitively reasonable. As a result, $y_c$ $(c = 1, \ldots, C)$ is generated from a Bernoulli distribution, i.e., $p(y_c|\pi_c) = (\pi_c)^{y_c}(1-\pi_c)^{(1-y_c)}$, where $\boldsymbol{\pi}$ is a probability vector based on a weighted empirical average of the Dirichlet realization $\boldsymbol{\lambda}^\top \bar{\mathbf{z}}$, $\bar{\mathbf{z}} = [\bar{z}_1, \ldots, \bar{z}_C]^\top$ is the average of $\mathbf{z}_1, \ldots, \mathbf{z}_M$: $\bar{z}_c = \frac{1}{M}\sum_{m=1}^M z_{mc}$. For example, $\boldsymbol{\pi}$ can be a Dirichlet distribution $\boldsymbol{\pi} \sim \text{Dir}(\lambda_1 \bar{z}_1, \ldots, \lambda_C \bar{z}_C)$. In this paper, we use a logistic model:

$$p(y_c = 1|\bar{\mathbf{z}}, \boldsymbol{\lambda}) = \frac{\exp(\lambda_c \bar{z}_c)}{1 + \exp(\lambda_c \bar{z}_c)}. \tag{1}$$

In practice, the set of possible instances can be quite diverse, such as pictures, texts, music and videos on a web page. Without loss of generality, we follow the convention of topic models to assume that each instance $\mathbf{x}$ is a *bag of discrete features* $\{f_1, f_2, \ldots, f_L\}$ and use a multinomial distribution[2]:

$$p(\mathbf{x}|\mathbf{z}, B) = p(\{f_1, \ldots, f_L\}|\mathbf{z}, B) \propto b_{c1}^{x_1} b_{c2}^{x_2} \ldots b_{cD}^{x_D}|_{z_c=1},$$

where $L$ is the total number of feature occurrences in $\mathbf{x}$ (e.g., the length of a paragraph), $B = [\mathbf{b}_1, \ldots, \mathbf{b}_D]$ is a $C \times D$-matrix with the $(c, d)$-th entry $b_{cd} = p(f_d = 1|z_c = 1)$ and $x_d$ is the frequency of $f_d$ in $\mathbf{x}$. The joint probability is then given by:

$$p(X, \mathbf{y}, Z, \boldsymbol{\theta}|\mathbf{a}, B, \boldsymbol{\lambda}) = p(\boldsymbol{\theta}|\mathbf{a}) \prod_{m=1}^{M} \left( p(\mathbf{z}_m|\boldsymbol{\theta}) \prod_{l=1}^{L} p(f_{ml}|B, \mathbf{z}_m) \right) p(\mathbf{y}|\bar{\mathbf{z}}, \boldsymbol{\lambda}). \qquad (2)$$

The graphical model for DBA is depicted in Figure 1(b). We can see that DBA has a diagram very similar to that of sLDA (Figure 1 in [2]). The key differences are: (1) Instead of using implicit topics for dimensionality reduction as in sLDA, DBA casts the predefined classes as explicit topics to discover the discriminative properties from the data; (2) A bag-of-feature instance rather than a single feature is generated conditioned on each sampled topic (class); (3) DBA models a multi-class, multi-label multi-instance corpus and can be applied directly to M³C, i.e., the classification of each pattern as well as the instances within it.

## 5 Parameter Estimation and Inference

Both parameter estimation and inferential tasks in DBA involve intractable computation of marginal probabilities. We use variational methods to approximate those distributions.

### 5.1 Variational Approximations

We use the following fully-factorized variational distribution to approximate the posterior distribution of the latent variables:

$$q(Z, \boldsymbol{\theta}|\boldsymbol{\gamma}, \Phi) = q(\boldsymbol{\theta}|\boldsymbol{\gamma}) \prod_{m=1}^{M} q(\mathbf{z}_m|\boldsymbol{\phi}_m) = \frac{\Gamma(\sum_{c=1}^{C} \gamma_c)}{\prod_{c=1}^{C} \Gamma(\gamma_c)} \prod_{c=1}^{C} \left( \theta_c^{\gamma_c - 1} \prod_{m=1}^{M} \phi_{mc}^{z_{mc}} \right), \qquad (3)$$

where $\boldsymbol{\gamma}$ and $\Phi = [\boldsymbol{\phi}_1, \ldots, \boldsymbol{\phi}_M]$ are variational parameters for a pattern $X$. We have:

$$\log P(X, \mathbf{y}|\mathbf{a}, B, \boldsymbol{\lambda}) = \log \int_{\boldsymbol{\theta}} \sum_{Z} p(X, \mathbf{y}, Z, \boldsymbol{\theta}|\mathbf{a}, B, \boldsymbol{\lambda}) d\boldsymbol{\theta}$$
$$= \mathcal{L}(\boldsymbol{\gamma}, \Phi) + KL(q(Z, \boldsymbol{\theta}|\boldsymbol{\gamma}, \Phi)||p(Z, \boldsymbol{\theta}|\mathbf{a}, B, \boldsymbol{\lambda})) \approx \max_{\boldsymbol{\gamma}, \Phi} \mathcal{L}(\boldsymbol{\gamma}, \Phi), \qquad (4)$$

where $KL(q(x)||p(x)) = \int_x q(x) \log \frac{q(x)}{p(x)} dx$ is the Kullback-Leibler (KL) divergence between two distributions $p$ and $q$, and $\mathcal{L}(\cdot)$ is the variational lower bound for the log-likelihood:

$$\mathcal{L}(\boldsymbol{\gamma}, \Phi) = \log \int_{\boldsymbol{\theta}} \sum_{Z} q(Z, \boldsymbol{\theta}|\boldsymbol{\gamma}, \Phi) \log \frac{p(X, \mathbf{y}, Z, \boldsymbol{\theta}|\mathbf{a}, B, \boldsymbol{\lambda})}{q(Z, \boldsymbol{\theta}|\boldsymbol{\gamma}, \Phi)} d\boldsymbol{\theta} = \mathbb{E}_q[\log p(\boldsymbol{\theta}|\mathbf{a})]$$
$$+ \sum_{m=1}^{M} \mathbb{E}_q[\log p(\mathbf{z}_m|\boldsymbol{\theta})] + \sum_{m=1}^{M} \mathbb{E}_q[\log p(\mathbf{x}_m|B, \mathbf{z}_m)] + \mathbb{E}_q[\log p(\mathbf{y}|\bar{\mathbf{z}}, \boldsymbol{\lambda})] + \mathcal{H}_q. \qquad (5)$$

The first two terms and the fifth term (the entropy of the variational distribution) in the right-hand side of Eq.(5) are identical to the corresponding terms in sLDA [2]. The third term, i.e., the variational expectation of the log likelihood for instance observations is:

$$\sum_{m=1}^{M} \mathbb{E}_q[\log p(\mathbf{x}_m|B, \mathbf{z}_m)] = \sum_{m=1}^{M} \sum_{c=1}^{C} \sum_{d=1}^{D} \phi_{mc} x_{md} \log b_{cd}. \tag{6}$$

The forth term in the righthand side of Eq.(5) corresponds to the expected log likelihood of observing the labels given the topic assignments:

$$\mathbb{E}_q[\log p(\mathbf{y}|\bar{\mathbf{z}}, \boldsymbol{\lambda})] = \frac{1}{M} \sum_{m=1}^{M} \sum_{c=1}^{C} (y_c - \frac{1}{2})\lambda_c \phi_{mc} - \sum_{c=1}^{C} \mathbb{E}_q[\log(\exp\frac{\lambda_c \bar{z}_c}{2} + \exp\frac{-\lambda_c \bar{z}_c}{2})]. \tag{7}$$

We bound the second term above by using the lower bound for logistic function [9]:

$$-\log(\exp\frac{\lambda_c \bar{z}_c}{2} + \exp\frac{-\lambda_c \bar{z}_c}{2}) \geqslant -\log(1 + \exp(-\xi_c)) - \frac{\xi_c}{2} + \varsigma_c(\lambda_c^2 \bar{z}_c^2 - \xi_c^2)$$
$$\approx -\log(1 + \exp(-\xi_c)) - \frac{\xi_c}{2} + 2\varsigma_c(\lambda_c \bar{z}_c \xi_c - \xi_c^2), \tag{8}$$

where $\boldsymbol{\xi}=[\xi_1,\dots,\xi_C]^\top$ are variational parameters, $\varsigma_c = \frac{1}{4\xi_c}\tanh(\frac{\xi_c}{2})$, and the second order residue term is omitted since the lower bound is exact when $\xi_c = -\lambda_c \bar{z}_c$.

Obtaining an approximate posterior distribution for the latent variables is then reduced to optimizing the objective $\max \mathcal{L}(q)$ or $\min KL(q||p)$ with respect to the variational parameters. By using Lagrange multipliers, we can easily derive the optimal condition which can be achieved by iteratively updating the variational parameters according to the following formulas:

$$\phi_{mc} \propto \prod_{d=1}^{D} (b_{cd})^{x_{md}} \exp\left(\Psi(\gamma_c) + \frac{\lambda_c}{2M}[2y_c - 1 + \tanh(\frac{\xi_c}{2})]\right),$$
$$\gamma_c = a_c + \sum_{m=1}^{M} \phi_{mc}, \qquad\qquad \xi_c = -\lambda_c \frac{1}{M} \sum_{m=1}^{M} \phi_{mc}, \tag{9}$$

where $\Psi(\cdot)$ is the digamma function. Note that instead of only one feature contributing to $\phi_{mc}$ as in LDA, all the features appearing in an instance are now responsible for contributing. This property tends to make DBA more robust to data sparsity. Also, DBA makes use of the supervision information with a term $\sum_{c=1}^{C} \lambda_c \bar{z}_c (2y_c - 1)$ in the variational likelihood bound $\mathcal{L}$. As $\mathcal{L}$ is optimized, this term is equivalent to maximizing the likelihood of sampling the classes to which the pattern belongs: $\{\max \lambda_c \sum_{m=1}^{M} z_{mc}, \text{ if } y_c = 1\}$ and simultaneously minimizing the likelihood of sampling the classes to which the pattern does not belong: $\{\min \lambda_c \sum_{m=1}^{M} z_{mc}, \text{ if } y_c = 0\}$. Here $\lambda_c$ (-$\lambda_c$) acts like a utility (cost) of assigning $X$ to the $c$-th class. As a result, it tends to align the Dirichlet topics discovered from the data to the class labels (Bernoulli observations) $\mathbf{y}$. This is why we coin the name **Dirichlet-Bernoulli Alignment**.

## 5.2 Parameter Estimation

The maximum likelihood parameter estimation of DBA relies on the variational approximation procedure. Given a corpus $\mathcal{D} = \{(X_n, \mathbf{y}_n)\}_{n=1,\dots,N}$, the MLE can be formulated as:

$$\mathbf{a}^*, B^*, \boldsymbol{\lambda}^* = \arg\max \log P(\mathcal{D}|\mathbf{a}, B, \boldsymbol{\lambda}) = \arg\max_{\mathbf{a},B,\boldsymbol{\lambda}} \sum_{n=1}^{N} \max_{\boldsymbol{\gamma}_n, \Phi_n} \mathcal{L}(\boldsymbol{\gamma}_n, \Phi_n|\mathbf{a}, B, \boldsymbol{\lambda}). \tag{10}$$

Table 1: Characteristic of the data sets.

| Data Set | #Train | #Test | $D$ | $C$ | $|Y|_{avg}$ | $\#(|Y| > 1)$ | $M_{avg}$ | $M_{min}$ | $M_{max}$ |
|----------|--------|-------|-----|-----|-------------|---------------|-----------|-----------|-----------|
| Text | 1200 | 679 | 500 | 10 | 1.4 | 721 (38.4%) | 8.2 | 1 | 36 |
| Query | 300 | 100 | 2000 | 101 | 1.4 | 99 (24.8%) | 65 | 3 | 731 |

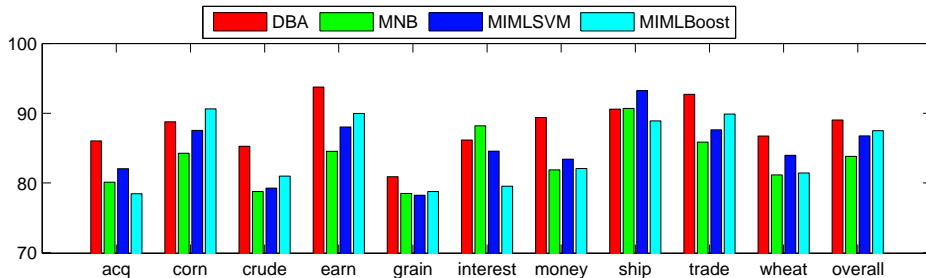

Figure 2: Accuracies(%) of DBA, MNB, MIMLSVM, and MIMLBoost for text classification.

The two-layer optimization in Eq.(10) involves two groups of parameters corresponding to the DBA model and its variational approximation, respectively. Optimizing alternatively between these two groups leads to a Variational Expectation Maximization (VEM) algorithm similar to the one used in LDA, where the E-step corresponds to the variational approximation for each pattern in the corpus. And the M-step in turn maximizes the objective in Eq.(6) w.r.t. the model parameters. These two steps are repeated alternatively until convergence.

### 5.3 Inference

DBA involves three types of inferential tasks. The first task is to infer the latent variables for a given pattern, which is straightforward after the variational approximation. The second task, pattern classification, addresses prediction of labels for a new pattern $X$: $p(y_c = 1|X; \mathbf{a}, B, \boldsymbol{\lambda}) \approx \exp(\lambda_c \bar{\phi}_c)/(1 + \exp(\lambda_c \bar{\phi}_c))$, where $\bar{\phi}_c = \frac{1}{M}\sum_{m=1}^{M} \phi_{mc}$ and the term $\frac{\lambda_c}{2M}[2y_c - 1 + \tanh(\frac{\xi_c}{2})]$ is removed when updating $\phi$ in Eq.(9). The third task, instance disambiguation, finds labels for each instances within a pattern: $p(\mathbf{z}_m|X, \mathbf{y}) = \int_{\boldsymbol{\theta}} p(\mathbf{z}_m, \boldsymbol{\theta}|X, \mathbf{y})d\boldsymbol{\theta} \approx q(\mathbf{z}_m|\phi_m)$, that is, $p(z_{mc} = 1|X, \mathbf{y}) = \phi_{mc}$.

## 6 Experiments

In this section, we conduct extensive experiments to test the DBA model as it is applied to pattern classification and instance disambiguation respectively. We first apply DBA to text classification and compare its performance with state-of-the-art M³C algorithms. Then the instance disambiguation performance of DBA is tested on a novel real-world task, i.e., named entity disambiguation for web search queries. Table 1 shows the information of the data sets used in our experiments.

### 6.1 Text Classification

This experiment is conducted on the `ModApte` split of the `Reuters-21578` text collection, which contains 10788 documents belonging to the most popular 10 classes. We use the top 500 words with the highest document frequency as features, and represent each document as a pattern with each of its paragraphs being an instance in order to exploit the semantic structure of documents explicitly. After eliminating the documents that have empty label set or less than 20 features, we obtain a subset of 1879 documents, among which 721 documents (about 38.4%) have multiple labels. The average number of labels per document is 1.4±0.6 and the average number of instances (paragraphs) per pattern (document) is 8.2±4.8. The data set is further randomly partitioned into a subset of 1200 documents for training and the rest for testing.

For comparison, we also test two state-of-the-art M³C algorithms, the *MIMLSVM* and *MIMLBoost* [13], and use the Multinomial Naïve Bayes (MNB) classifier trained on the vector space model of the whole documents as the baseline. For a fair comparison, linear kernel is used in both MIMLSVM and MIMLBoost and all the hyper-parameters are tuned by 5-fold cross validation prior to training. We use the Hamming-Accuracy [13] to evaluate the results, for DBA and MNB, the label is estimated by: $y = \delta(p(y = 1|X) \geqslant t)$, where the cut-off probability threshold is also selected based on 5-fold cross validation. Each experiment is repeated for 5 random runs and the average results are reported by a bar chart as depicted in Figure 2. We can see that: (1) for most classes, the three

Table 2: Accuracy@$N$ ($N = 1, 2, 3$) and micro-averaged and macro-averaged F-measures of DBA, MNB and SVM based disambiguation methods.

| Method | A@1 | *Gain* | A@2 | *Gain* | A@3 | *Gain* | $\mathbf{F}_{micro}$ | *Gain* | $\mathbf{F}_{macro}$ | *Gain* |
|---|---|---|---|---|---|---|---|---|---|---|
| MNB-TF | 0.4154 | 30.4% | 0.4913 | 25.7% | 0.5168 | 25.4% | 0.4154 | 30.4% | 0.3144 | 47.0% |
| MNB-TF-IDF | 0.4177 | 29.6% | 0.4918 | 25.6% | 0.5176 | 25.2% | 0.4177 | 29.6% | 0.2988 | 54.7% |
| SVM-TF | 0.4927 | 9.9% | NA | | NA | | 0.4927 | 9.9% | 0.3720 | 24.2% |
| SVM-TF-IDF | 0.4912 | 10.2% | NA | | NA | | 0.4912 | 10.2% | 0.3670 | 25.0% |
| DBA | 0.5415 | - | 0.6175 | - | 0.6482 | - | 0.5415 | - | 0.4622 | |

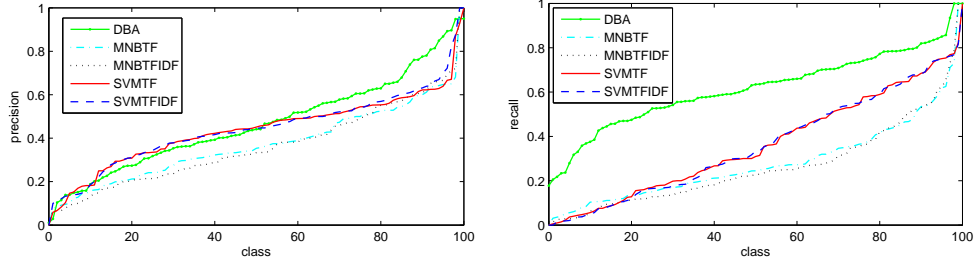

Figure 3: Precision and Recall scores for each of 101 classes by using DBA, MNB and SVM based methods.

M$^3$C algorithms outperform the MNB baseline; (2) the performance of DBA is at least comparable with MIMLBoost and MIMLSVM. For most classes and *overall*, DBA performs the best, whereas for some classes, MIMLBoost and MIMLSVM perform even slightly worse than MNB. A possible reason might be: if the documents are very short, splitting them might introduce severe data sparseness and in turn harms the performance. We also observe that DBA is much more efficient than MIMLBoost and MIMLSVM. For training, DBA takes 42 mins on average, in contrast to 557 minutes (MIMLSVM) and 806 minutes (MIMLBoost).

## 6.2 Named Entity Disambiguation

Query ambiguity is a fundamental obstacle for search engine to capture users' search intentions. In this section, we employ DBA to disambiguate the named entities in web search queries. This is a very challenging problem because queries are usually very short (2 to 3 words on average), noisy (e.g., misspellings, abbreviations, less grammatical structure) and topic-distracted. A single named-entity query $Q$ can be viewed as a combination of a single named entity $e$ and a set of context words $\mathbf{w}$ (the remaining text in $Q$). By differentiating the possible meanings of the named entity in a query and identifying the most possible one, entity disambiguation can help search engines to capture the precise information need of the user and in turn improve search by responding with the truly most relevant documents. For example, when a user inputs "*When are the casting calls for Harry Potter in USA?*", the system should be able to identify that the ambiguous named entity "*Harry Potter*" (i.e., it can be a *movie*, a *book* or a *game*) really refers to a *movie* in this specific query.

We treat the ambiguity of $e$ as a hidden class $z$ over $e$ and make use of the query log as a data source for mining the relationship among $e$, $\mathbf{w}$ and $z$. In particular, the query log can be viewed as a multi-class, multi-label and multi-instance corpus $\{(X_n, Y_n)\}_{n=1,2,...,N}$, in which each pattern $X$ corresponds to a named-entity $e$ and is characterized by a set of instances $\{\mathbf{x}_m\}_{m=1,2,...,M}$ corresponding to all the contexts $\{\mathbf{w}_m\}_{m=1,2,...,M}$ that co-occur with $e$ in queries, and the label $Y$ contains all the ambiguities of $e$.

Our data was based on a snapshot of `answers.yahoo.com` crawled in early 2008, containing 216563 queries from 101 classes. We manually collect 400 named entities and label them according to the labels of their co-occurring queries in Yahoo! CQA. A randomly chosen subset of 300 entities are used as training data and the other 100 are used for testing. We compare our DBA based method with baselines including Multinomial Naïve Bayes classifier using TF (*MNB-TF*) or TF-IDF (*MNB-TFIDF*) as word attributes, and SVM classifier using TF (*SVM-TF*) or TFIDF (*SVM-TF-IDF*). For SVM, a similar scheme as MIMLSVM is used for learning M$^3$C classifiers.

Table 2 demonstrates the Accuracy@$N$ ($N = 1, 2, 3$) as well as micro-averaged and macro-average F-measure scores of each disambiguation approach[3]. All the results are obtained through 5-fold cross-validation. From the table, we observe that DBA achieves significantly better performance than all the other methods. In particular, for Accuracy@1 scores, DBA can achieve a gain of about

30% relative to two MNB methods, and about $10\%$ relative to two SVM methods; for macro-average F-measures, DBA can achieve a gain of about $50\%$ over MNB methods, and about $25\%$ over SVM methods. As a reference, in Figure 3, we also illustrate the sorted precision and recall scores for each of the 101 classes. We can see that, DBA slightly outperforms the baselines in terms of precision, and significantly performs better in terms of the recall scores. In particular, for recall, DBA can achieve a gain of more than $50\%$ relative to MNB and SVM baselines.

## 7    Concluding Remarks

Multi-class, multi-label and multi-instance classification ($M^3C$) is encountered in many applications. Even for task that is not explicitly an $M^3C$ problem, it might still be advantageous to treat it as $M^3C$ so as to better explore its inner structures and effectively handle the ambiguities. $M^3C$ also naturally arises from the difficulty of acquiring finely-labeled data. In this paper, we have proposed a probabilistic generative model for $M^3C$ corpora. The proposed DBA model is useful for both pattern classification and instance disambiguation, as has been tested respectively in text classification and named-entity disambiguation tasks.

An interesting observation in practice is that, although there might be a large number of classes/topics, usually a pattern is only associated with a very limited number of them. In our experiment, we found that substantial improvement could be achieved by simply enforcing label sparsity, e.g., by using LASSO style regularization. In future, we will investigate such "Label Parsimoniousness" in a principled way. Another meaningful investigation would be to explicitly capture or explore the class correlations by using, for example, the Logistic Normal distribution [3] rather than Dirichlet.

### Acknowledgments

Hongyuan Zha is supported by NSF #DMS-0736328 and grant from Microsoft. Bao-Gang Hu is supported by NSFC #60275025 and the MOST of China grant #2007DFC10740.

## Footnotes

[1] A "pattern" or "example" is a typical sample in a data collection and an "instance" is a part of a "pattern".

[2]This is only a simple special case instance model for DBA. It is quite straightforward to substitute other instance models such as Gaussian, Poisson and other more complicated models like Gaussian mixtures.

[3]Since SVM only outputs hard class assignments, there is no Accuracy@2,3 for SVM based methods.

### References

[1]  Andrews S. and Hofmann T. (2003) Multiple Instance Learning via Disjunctive Programming Boosting, In *Advances in Neural Information Processing Systems 17* (*NIPS'03*), MIT Press.

[2]  Blei D. and McAuliffe J. (2007) Supervised topic models. In *Advances in Neural Information Processing Systems 21* (*NIPS'07*), MIT Press.

[3]  Blei D. and Lafferty J. (2007) A correlated topic model of Science. *Annals of Applied Statistics*. Vol. 1, No. 1, pp. 17–35, 2007.

[4]  Blei D., Ng A. and Jordan M. (2003) Latent Dirichlet Allocation. *Journal of Machine Learning Research*, Vol. 3, pp.993–1022, Jan. 2003, MIT Press.

[5]  Boutell M. R., Luo J., Shen X. and Brown C. M. (2004) Learning Multi-Label Scene Classification. *Pattern Recognition*, 37(9), pp.1757–1771, 2004.

[6]  Cour T., Sapp B., Jordan C. and Taskar B. (2009) Learning from Ambiguously Labeled Images, In the 23rd IEEE Conference on Computer Vision and Pattern Recognition (*CVPR'09*).

[7]  Dietterich T. G., Lathrop R. H., Lozano-Perez T. (1997) Solving the Multiple-Instance Problem with Axis-Parallel Rectangles. *Artificial Intelligence Journal*, Vol. 89, pp.31–71, Jan.1997.

[8]  Ghamrawi N. and McCallum A. (2005) Collective Multi-Label Classification, In *ACM International Conference On Information And Knowledge Management (CIKM'05)*, pp.195–200.

[9]  Jaakkola, T. and Jordan M. I. (2000). Bayesian parameter estimation via variational methods. *Statistics and Computing*, Vol 10, Issue 1, pp. 25–37.

[10]  Ueda N. and Saito K. (2002) Parametric Mixture Models For Multi-Labeled Text. In *Advances in Neural Information Processing Systems 15* (*NIPS'02*).

[11]  Viola P., Platt J. and Zhang C. (2006). Multiple Instance Boosting For Object Detection. In *Advances in Neural Information Processing Systems 20* (*NIPS'06*), pp.1419–1426, MIT Press.

[12]  Xu G., Yang S.-H. and Li H. (2009) Named Entity Mining from Click-Through Data Using Weakly Supervised LDA, In ACM Knowledge Discovery and Data Mining (*KDD'09*).

[13]  Zhou Z.-H. and Zhang M.-L. (2006) Multi-Instance Multi-Label Learning with Application to Scene Classification, In *Advances in Neural Information Processing Systems 20* (*NIPS'06*).

